# On the Sample Complexity of Robust PCA

**Matthew Coudron**
Department of Electrical Engineering and Computer Science
Massachusetts Institute of Technology
Cambridge, MA 02139
mcoudron@mit.edu

**Gilad Lerman**
School of Mathematics
University of Minnesota
Minneapolis, MN 55455
lerman@umn.edu

## Abstract

We estimate the rate of convergence and sample complexity of a recent robust estimator for a generalized version of the inverse covariance matrix. This estimator is used in a convex algorithm for robust subspace recovery (i.e., robust PCA). Our model assumes a sub-Gaussian underlying distribution and an i.i.d. sample from it. Our main result shows with high probability that the norm of the difference between the generalized inverse covariance of the underlying distribution and its estimator from an i.i.d. sample of size $N$ is of order $O(N^{-0.5+\epsilon})$ for arbitrarily small $\epsilon > 0$ (affecting the probabilistic estimate); this rate of convergence is close to the one of direct covariance estimation, i.e., $O(N^{-0.5})$. Our precise probabilistic estimate implies for some natural settings that the sample complexity of the generalized inverse covariance estimation when using the Frobenius norm is $O(D^{2+\delta})$ for arbitrarily small $\delta > 0$ (whereas the sample complexity of direct covariance estimation with Frobenius norm is $O(D^2)$). These results provide similar rates of convergence and sample complexity for the corresponding robust subspace recovery algorithm. To the best of our knowledge, this is the only work analyzing the sample complexity of any robust PCA algorithm.

## 1 Introduction

A fundamental problem in probability and statistics is to determine with overwhelming probability the rate of convergence of the empirical covariance (or inverse covariance) of an i.i.d. sample of increasing size $N$ to the covariance (or inverse covariance) of the underlying random variable (see e.g., [17, 3] and references therein). Clearly, this problem is also closely related to estimating with high probability the sample complexity, that is, the number of samples required to obtain a given error of approximation $\epsilon$. In the case of a compactly supported (or even more generally sub-Gaussian) underlying distribution, it is a classical exercise to show that this rate of convergence is $O(N^{-0.5})$ (with a comparability constant depending on properties of $\mu$, in particular $D$, as well as on the threshold probability, see e.g., [17, Proposition 2.1]). The precise estimate for this rate of convergence implies that the sample complexity of covariance estimation is $O(D)$ when using the spectral norm and $O(D^2)$ when using the Frobenius norm. The rate of convergence and sample complexity of PCA immediately follow from these estimates (see e.g., [15]).

While such estimates are theoretically fundamental, they can be completely useless in the presence of outliers. That is, direct covariance or inverse covariance estimation and its resulting PCA are very sensitive to outliers. Many robust versions of covariance estimation, PCA and dimension reduction have been developed in the last three decades (see e.g., the standard textbooks [8, 10, 14]). In the last few years new convex algorithms with provable guarantees have been suggested for robust subspace recovery and its corresponding dimension reduction [5, 4, 19, 20, 11, 7, 2, 1, 21, 9].

Most of these works minimize a mixture of an $\ell_1$-type norm (depending on the application) and the nuclear norm. Their algorithmic complexity is not as competitive as PCA and their sample com-

plexity is hard to estimate due to the problem of extending the nuclear norm out-of-sample. On the other hand, Zhang and Lerman [21] have proposed a novel M-estimator for robust PCA, which is based on a convex relaxation of the sum of Euclidean distances to subspaces (which is originally minimized over the non-convex Grassmannian). This procedure suggests an estimator for a generalized version of the inverse covariance matrix and uses it to robustly recover an underlying low-dimensional subspace. This idea was extended in [9] to obtain an even more accurate method for subspace recovery, though it does not estimate the generalized inverse covariance matrix (in particular, it has no analogous notion of singular values or their inverses). The algorithmic complexity of the algorithms solving the convex formulations of [21] and [9] is comparable to that of full PCA. Here we show that for the setting of sub-Gaussian distributions the sample complexity of the robust PCA algorithm in [21] (or its generalized inverse covariance estimation) is close to that of PCA (or to sample covariance estimation). Our analysis immediately extends to the robust PCA algorithm of [9].

## 1.1 The Generalized Inverse Covariance and its Corresponding Robust PCA

Zhang and Lerman [21] formed the set

$$\mathbb{H} := \{\mathbf{Q} \in \mathbb{R}^{D \times D} : \mathbf{Q} = \mathbf{Q}^T, \operatorname{tr}(\mathbf{Q}) = 1\}, \tag{1.1}$$

as a convex relaxation for the orthoprojectors (from $\mathbb{R}^D$ to $\mathbb{R}^D$), and defined the following energy function on $\mathbb{H}$ (with respect to a data set $\mathcal{X}$ in $\mathbb{R}^D$):

$$F_{\mathcal{X}}(\mathbf{Q}) := \sum_{\mathbf{x} \in \mathcal{X}} \|\mathbf{Q}\mathbf{x}\|, \tag{1.2}$$

where $\|\cdot\|$ denotes the Euclidean norm of a vector in $\mathbb{R}^D$. Their generalized empirical inverse covariance is

$$\hat{\mathbf{Q}}_{\mathcal{X}} = \underset{\mathbf{Q} \in \mathbb{H}}{\arg\min}\, F_{\mathcal{X}}(\mathbf{Q}). \tag{1.3}$$

They showed that when replacing the term $\|\mathbf{Q}\mathbf{x}\|$ by $\|\mathbf{Q}\mathbf{x}\|^2$ in (1.2) and when $\operatorname{Sp}\{\mathcal{X}\} = \mathbb{R}^D$, then the minimization (1.3) results in a scaled version of the empirical inverse covariance matrix. It is thus clear why we can refer to $\hat{\mathbf{Q}}_{\mathcal{X}}$ as a generalized empirical inverse covariance (or $\ell_1$-type version of it). We describe the absolute notion of generalized inverse covariance matrix, i.e., non-empirical, in §1.2. Zhang and Lerman [21] did not emphasize the empirical generalized inverse covariance, but the robust estimate of the underlying low-dimensional subspace by the span of the bottom eigenvectors of this matrix. They rigorously proved that such a procedure robustly recovers the underlying subspace under some conditions.

## 1.2 Main Result of this Paper

We focus on computing the sample complexity of the estimator $\hat{\mathbf{Q}}_{\mathcal{X}}$. This problem is practically equivalent with estimating the rate of convergence of $\hat{\mathbf{Q}}_{\mathcal{X}}$ of an i.i.d. sample $\mathcal{X}$ to the "generalized inverse covariance" of the underlying distribution $\mu$. We may assume that $\mu$ is a sub-Gaussian probability measure on $\mathbb{R}^D$ (see §2.1 and the extended version of this paper). However, in order to easily express the dependence of our probabilistic estimates on properties of the measure $\mu$, we assume for simplicity that $\mu$ is compactly supported and denote by $R_\mu$ the minimal radius among all balls containing the support of $\mu$, that is,

$$R_\mu = \min\{r > 0 : \operatorname{supp}(\mu) \subseteq B(\mathbf{0}, r)\},$$

where $B(\mathbf{0}, r)$ is the ball around the origin $\mathbf{0}$ with radius $r$. We further assume that for some $0 < \gamma < 1$, $\mu$ satisfies the following condition, which we refer to as the "two-subspaces criterion" (for $\gamma$): For any pair of $(D-1)$-dimensional subspaces of $\mathbb{R}^D$, $L_1$ and $L_2$:

$$\mu((L_1 \cup L_2)^c) \geq \gamma. \tag{1.4}$$

We note that if $\mu$ satisfies the two-subspaces criterion for any particular $0 < \gamma < 1$, then its support cannot be a union of two hyperplanes of $\mathbb{R}^D$. The use of this assumption is clarified below in §3.2, though it is possible that one may weaken it.

We first formulate the generalized inverse covariance of the underlying measure as follows:

$$\hat{\mathbf{Q}} = \arg \min_{\mathbf{Q} \in \mathbb{H}} F(\mathbf{Q}), \tag{1.5}$$

where

$$F(\mathbf{Q}) = \int \|\mathbf{Q}\mathbf{x}\| \, d\mu(\mathbf{x}). \tag{1.6}$$

Let $\{\mathbf{x}_i\}_{i=1}^{\infty}$ be a sequence of i.i.d. random variables sampled from $\mu$ (i.e., each variable has distribution $\mu$). Let $\mathcal{X}_N := \{\mathbf{x}_i\}_{i=1}^{N}$ and denote

$$\hat{\mathbf{Q}}_N := \hat{\mathbf{Q}}_{\mathcal{X}_N} \quad \text{and} \quad F_N := F_{\mathcal{X}_N}. \tag{1.7}$$

Our main result shows with high probability that $\hat{\mathbf{Q}}$ and $\hat{\mathbf{Q}}_N$ are uniquely defined (which we denote by u.d. from now on) and that $\{\hat{\mathbf{Q}}_N\}_{N \in \mathbb{N}}$ converges to $\hat{\mathbf{Q}}$ in the following specified rate. It uses the common notation: $a \vee b := \max(a, b)$. We explain its implications in §2.

**Theorem 1.1.** *If $\mu$ is a compactly supported distribution satisfying the two-subspaces criterion for $\gamma > 0$, then there exists a constant $\alpha_0 \equiv \alpha_0(\mu, D, \epsilon) > 0$ such that for any $\epsilon > 0$ and $N > 2(D-1)$ the following estimate holds:*

$$\mathbb{P}\left( \hat{\mathbf{Q}} \,\&\, \hat{\mathbf{Q}}_N \text{ are u.d. and } \|\hat{\mathbf{Q}} - \hat{\mathbf{Q}}_N\|_F \leq \frac{2}{\alpha_0} N^{-\frac{1}{2}+\epsilon} \right)$$

$$\geq 1 - C_0 N^{D^2} \exp\left( \frac{-N^{2\epsilon}}{D \cdot R_\mu^2} \right) - 2 \binom{N}{D-1}^2 (1-\gamma)^{N-2(D-1)}, \tag{1.8}$$

*where*

$$C_0 \equiv C_0(\alpha_0, D) := 4 \cdot ((4\alpha_0) \vee 2) \cdot \left( 10 D \frac{2\alpha_0 + 4((4\alpha_0) \vee 2) R_\mu}{1 - \frac{2\alpha_0}{(4\alpha_0) \vee 2}} \right)^{\frac{D(D+1)}{2}}. \tag{1.9}$$

Intuitively, $\alpha_0$ represents a lower bound on the directional second derivatives of $F$. Therefore, $\alpha_0$ should affect sample complexity because the number of random samples taken to approximate a minimum of $F$ should be affected by how sharply $F$ increases about its minimum. It is an interesting and important open problem to find lower bounds on $\alpha_0$ for general $\mu$.

## 2    Implication and Extensions of the Main Result

### 2.1    Generalization to Sub-Gaussian Measures

We can remove the assumption that the support of $\mu$ is bounded (with radius $R_\mu$) and assume instead that $\mu$ is sub-Gaussian. In this case, instead of Hoeffding's inequality, we apply [18, Proposition 5.10] with $a_i = 1$ for all $1 \leq i \leq n$. When formulating the corresponding inequality, one may note that $\sup_{p \geq 1} p^{-1/2} (E_\mu |\mathbf{x}|^p)^{1/p}$ (where $\mathbf{x}$ represents a random variable sampled from $\mu$) can be regarded as a substitute for $R_\mu$ (see [21] for more details of a similar analysis).

### 2.2    Sample Complexity

The notion of sample complexity arises in the framework of Probably-Approximately-Correct Learning of Valiant [16]. Generally speaking, the sample complexity in our setting is the minimum number of samples $N$ required, as a function of dimension $D$, to achieve a good estimation of $\hat{\mathbf{Q}}$ with high probability. We recall that in this paper we use the Frobenius norm for the estimation error. The following calculation will show that under some assumptions on $\mu$ it suffices to use $N = \Omega(D^\eta)$ samples for any $\eta > 2$ (we recall that $f(x) = \Omega(g(x))$ as $x \to \infty$ if and only if $g(x) = O(f(x))$). In our analysis we will have to assume that $\gamma$ is a fixed constant, and $\alpha_0$ goes as $1/\sqrt{D}$. These assumptions are placing additional restrictions on the measure $\mu$, which we expect to be reasonable in practice as we later clarify. We further assume that $R_\mu = O(D^{-0.5})$ and also explain later why it makes sense for the setting of robust subspace recovery.

To bound the sample complexity we set $C_1 := 4 \cdot ((4\alpha_0) \vee 2)$ and $C_2 := 10 \cdot (2\alpha_0 + 4((4\alpha_0) \vee 2)R_\mu)/(1 - 2\alpha_0/(4\alpha_0) \vee 2)$ so that $C_0 \leq C_1 \cdot (C_2 \cdot D)^{D^2}$ (see (1.9)). Applying this bound and (1.8) we obtain that if $\eta > 2$ is fixed, $1/\eta < \epsilon < \frac{1}{2}$ and $N \geq D^\eta$, then

$$\mathbb{P}\left(\hat{\mathbf{Q}} \,\&\, \hat{\mathbf{Q}}_N \text{ are u.d. and } \|\hat{\mathbf{Q}} - \hat{\mathbf{Q}}_N\|_F \leq \frac{2}{\alpha_0}N^{-\frac{1}{2}+\epsilon}\right) \tag{2.1}$$

$$\geq 1 - C_1(C_2 \cdot D \cdot N)^{D^2} \exp\left(\frac{-N^{2\epsilon}}{D \cdot R_\mu^2}\right) - 2\,N^{2(D-1)}(1-\gamma)^{N-2(D-1)}$$

$$\geq 1 - C_1 \exp\left(\log(C_2 \cdot D^{1+\eta})D^2 - D^{2\eta\epsilon}\right)$$
$$- 2\exp\left(2\eta(D-1)\log(D) + \log(1-\gamma)(D^\eta - 2(D-1))\right).$$

Since $\epsilon > 1/\eta$ the first term in the RHS of (2.1) decays exponentially as a function of $D$ (or, equivalently, as a function of $N \geq D^\eta$). Similarly, since $0 < \gamma < 1$ and $\eta > 1$ the second term in the RHS of (2.1) decays exponentially as a function of $D$. Furthermore, since $\epsilon < \frac{1}{2}$ it follows that the error term for the minimizer, i.e., $N^{-\frac{1}{2}+\epsilon} \leq D^{\eta(\epsilon-\frac{1}{2})}$, decays polynomially in $D$. Thus, in order to achieve low error estimation with high probability it is sufficient to take $N = \Omega(D^\eta)$ samples for any $\eta > 2$. The exact guarantees on error estimation and probability of error can be manipulated by changing the constant hidden in the $\Omega$ term.

We would like to point out the expected tradeoff between the sample complexity and the rate of convergence. If $\epsilon$ approaches $0$, then the rate of convergence becomes optimal but the sample complexity deteriorates. On the other hand, if $\epsilon$ approaches $0.5$, then the sample complexity becomes optimal, but the rate of convergence deteriorates.

To motivate our assumption on $R_\mu$, $\gamma$ and $\alpha_0$, we recall the needle-haystack and syringe-haystack models of [9] as a prototype for robust subspace recovery. These models assume a mixtures of outlier and inliers components. The distribution of the outliers component is normal $N(\mathbf{0}, (\sigma_{\text{out}^2}/D)\mathbf{I}_D)$ and the distribution of the inliers component is a mixture of $N(\mathbf{0}, (\sigma_{\text{in}^2}/d)\mathbf{P}_L)$ (where $L$ is a $d$-subspace) and $N(\mathbf{0}, (\sigma_{\text{in}^2}/(CD))\mathbf{I}_D)$, where $C \gg 1$ (the latter component has coefficient zero in the needle-haystack model).

The underlying distribution of the syringe-haystack (or needle-haystack) model is not compactly supported, but clearly sub-Gaussian (as discussed in §2.1) and its standard deviation is of order $O(D^{-0.5})$. We also note that $\gamma$ here is the coefficient of the outlier component in the needle-haystack model, which we denote by $\nu_0$. Indeed, the only non-zero measure that can be contained in a (D-1)-dimensional subspace is the measure associated with $N(\mathbf{0}, (\sigma_{\text{in}^2}/d)\mathbf{P}_L)$, and that has total weight at most $(1 - \nu_0)$. It is also possible to verify explicitly that $\alpha_0$ is lower bounded by $1/\sqrt{D}$ in this case (though our argument is currently rather lengthy and will appear in the extended version of this paper).

## 2.3 From Generalized Covariances to Subspace Recovery

We recall that the underlying $d$-dimensional subspace can be recovered from the bottom $d$ eigenvectors of $\hat{\mathbf{Q}}_N$. Therefore, the rate of convergence of the subspace recovery (or its corresponding sample complexity) follows directly from Theorem 1.1 and the Davis-Kahan Theorem [6]. To formulate this, we assume here for simplicity that $\hat{\mathbf{Q}}$ and $\hat{\mathbf{Q}}_N$ are u.d. (recall Theorems 3.1 and 3.2).

**Theorem 2.1.** *If $d < D$, $\epsilon > 0$, $\alpha_0 \equiv \alpha_0(\mu, D, \epsilon)$ is the positive constant guaranteed by Theorem 2.1, $\hat{\mathbf{Q}}$ and $\hat{\mathbf{Q}}_N$ are u.d. and $\hat{L}_d$, $\hat{L}_{d,N}$ are the subspaces spanned by the bottom $d$ eigenvectors (i.e., with lowest $d$ eigenvalues) of $\hat{\mathbf{Q}}$ and $\hat{\mathbf{Q}}_N$ respectively, $\mathbf{P}_{\hat{L}_d}$ and $\mathbf{P}_{\hat{L}_{d,N}}$ are the orthoprojectors on these subspaces and $\nu_{D-d}$ is the $(D-d)$th eigengap of $\hat{\mathbf{Q}}$, then*

$$\mathbb{P}\left(\|\mathbf{P}_{\hat{L}_d} - \mathbf{P}_{\hat{L}_{d,N}}\|_F \leq \frac{4}{\alpha_0 \cdot \nu_{D-d}}N^{-\frac{1}{2}+\epsilon}\right) \geq 1 - C_0 N^{D^2} \exp\left(\frac{-N^{2\epsilon}}{D \cdot R_\mu^2}\right). \tag{2.2}$$

## 2.4 Nontrivial Robustness to Noise

We remark that (2.2) implies nontrivial robustness to noise for robust PCA. Indeed, assume for example an underlying $d$-subspace $L_d^*$ and a mixture distribution (representing noisy inliers/outliers

components) whose inliers component is symmetric around $L_d^*$ with relatively high level of variance in the orthogonal component of $\hat{L}_d$ and its outliers component is spherically symmetric with sufficiently small mixture coefficient. One can show that in this case $\hat{L}_d = L_d^*$. Combining this observation and (2.2), we can verify robustness to nontrivial noise when recovering $L_d^*$ from i.i.d. samples of such distributions.

## 2.5 Convergence Rate of the REAPER Estimator

The REAPER and S-REAPER Algorithms [9] are variants of the robust PCA algorithm of [21]. The objective of the REAPER algorithm can be formulated as aiming to minimize the energy $F_{\mathcal{X}}(\mathbf{Q})$ over the set

$$\mathbb{G} := \{\mathbf{Q} \in \mathbb{R}^{D \times D} : \mathbf{Q} = \mathbf{Q}^T, \text{tr}(\mathbf{Q}) = D - d \text{ and } \mathbf{Q} \preccurlyeq I\}, \tag{2.3}$$

where $\preccurlyeq$ denotes the semi-definite order. The $d$-dimensional subspace can then be recovered by the bottom $d$ eigenvectors of $\mathbf{Q}$ (in [9] this minimization is formulated with $\mathbf{P} = I - \mathbf{Q}$, whose top $d$ eigenvectors are found). The rate of convergence of the minimizer of $F_{\mathcal{X}}(\mathbf{Q})$ over $\mathbb{G}$ to the minimizer of $F(\mathbf{Q})$ over $\mathbb{G}$ is similar to that in Theorem 1.1. The proof of Theorem 1.1 must be modified to deal with the boundary of the set $\mathbb{G}$. If the minimizer $\hat{\mathbf{Q}}$ lies on the interior of $\mathbb{G}$ then the proof is the same. If $\hat{\mathbf{Q}}$ is on the boundary of $\mathbb{G}$ we must only consider the directional derivatives which point towards the interior of $\mathbb{G}$, or tangent to the boundary. Other than that the proof is the same.

## 2.6 Convergence Rate with Additional Sparsity Term

Rothman et al. [13] and Ravikumar et al. [12] have analyzed an estimator for sparse inverse covariance. This estimator minimizes over all $\mathbf{Q} \succ 0$ the energy

$$\langle \mathbf{Q}, \widehat{\boldsymbol{\Sigma}}_N \rangle_F - \log \det(\mathbf{Q}) + \lambda_N \|\mathbf{Q}\|_{\ell_1}, \tag{2.4}$$

where $\widehat{\boldsymbol{\Sigma}}_N$ is the empirical covariance matrix based on sample of size $N$, $\langle \cdot, \cdot \rangle_F$ is the Frobenius inner product (i.e., sum of elementwise products) and $\|\mathbf{Q}\|_{\ell_1} = \sum_{i,j=1}^D |\mathbf{Q}_{i,j}|$.

Zhang and Zou [22] have suggested a similar minimization, which replaces the first two terms in (2.4) (corresponding to $\lambda_N = 0$) with

$$\langle \mathbf{Q}^2, \widehat{\boldsymbol{\Sigma}}_N \rangle_F / 2 - \text{tr}(\mathbf{Q}). \tag{2.5}$$

Indeed, the minimizers of (2.4) when $\lambda_N = 0$ and of (2.5) are both equal to $\widehat{\boldsymbol{\Sigma}}_N^{-1}$ (assuming that the $\text{Sp}(\{\mathbf{x}_i\}_{i=1}^N) = \mathbb{R}^D$ so that the inverse empirical covariance exists).

Using the definition of $\widehat{\boldsymbol{\Sigma}}_N$, i.e., $\widehat{\boldsymbol{\Sigma}}_N = \sum_{i=1}^N \mathbf{x}_i \mathbf{x}_i^T / N$, we note that

$$\langle \mathbf{Q}^2, \widehat{\boldsymbol{\Sigma}}_N \rangle_F = \frac{1}{N} \sum_{i=1}^N \|\mathbf{Q}\mathbf{x}_i\|^2. \tag{2.6}$$

Therefore, the minimizer of (2.5) over all $\mathbf{Q} \succ 0$ is the same up to a multiplicative constant as the minimizer of the RHS of (2.6) over all $\mathbf{Q} \succ 0$ with $\text{tr}(\mathbf{Q}) = 1$. Teng Zhang suggested to us replacing the RHS of (2.6) with $F_{\mathcal{X}}$ and modifying the original problem of (2.4) (or more precisely its variant in [22]) with the minimization over all $\mathbf{Q} \in \mathbb{H}$ of the energy

$$F_{\mathcal{X}}(\mathbf{Q}) + \lambda_N \|\mathbf{Q}\|_{\ell_1}. \tag{2.7}$$

The second term enforces sparseness and we expect the first term to enforce robustness.

By choosing $\lambda_N = O(N^{-0.5})$ we can obtain similar rates of convergence for the minimizer of (2.7) as the one when $\lambda_N = 0$ (see extended version of this paper), namely, rate of convergence of order $O(N^{-0.5+\epsilon})$ for any $\epsilon > 0$. The dependence on $D$ is also the same. That is, the minimum sample size when using the Frobenius norm is $O(D^\eta)$ for any $\eta > 2$. Nevertheless, Ravikumar et al. [12] show that under some assumptions (see e.g., Assumption 1 in [12]), the minimal sample size is $O(\log(D)r^2)$, where $r$ is the maximum node degree for a graph, whose edges are the nonzero entries of the inverse covariance. It will be interesting to generalize such estimates to the minimization of (2.7).

# 3 Overview of the Proof of Theorem 1.1

## 3.1 Structure of the Proof

We first discuss in §3.2 conditions for uniqueness of $\hat{\mathbf{Q}}$ and $\hat{\mathbf{Q}}_N$ (with high probability). In §3.3 and §3.4 we explain in short the two basic components of the proof of Theorem 1.1. The first of them is that $\|\hat{\mathbf{Q}} - \hat{\mathbf{Q}}_N\|_F$ can be controlled from above by differences of directional derivatives of $F$. The second component is that the rate of convergence of the derivatives of $\{F_N\}_{N=1}^\infty$ to the derivative of $F$ is easily obtained by Hoeffding's inequality. In §3.5 we gain some intuition for the validity of Theorem 1.1 in view of these two components and also explain why they are not sufficient to conclude the proof. In §3.6 we describe the construction of "nets" of increasing precision; using these nets we conclude the proof of Theorem 1.1 in §3.7. Throughout this section we only provide the global ideas of the proof, whereas in the extended version of this paper we present the details.

## 3.2 Uniqueness of the Minimizers

The two-subspaces criterion for $\mu$ guarantees that $\hat{\mathbf{Q}}$ is u.d. and that $\hat{\mathbf{Q}}_N$ is u.d. with overwhelming probability for sufficiently large $N$ as follows.

**Theorem 3.1.** *If $\mu$ satisfies the two-subspaces criterion for some $\gamma > 0$, then $F$ is strictly convex.*

**Theorem 3.2.** *If $\mu$ satisfies the two-subspaces criterion for some $\gamma > 0$ and $N > 2(D-1)$, then*

$$\mathbb{P}\left(F_N \text{ is not strictly convex}\right) \leq 2 \binom{N}{D-1}^2 (1-\gamma)^{N-2(D-1)}. \tag{3.1}$$

## 3.3 From Energy Minimizers to Directional Derivatives of Energies

We control the difference $\|\mathbf{Q} - \hat{\mathbf{Q}}\|_F$ from above by differences of derivatives of energies at $\mathbf{Q}$ and $\hat{\mathbf{Q}}$. Here $\mathbf{Q}$ is an arbitrary matrix in $B_r(\hat{\mathbf{Q}})$ for some $r > 0$ (where $B_r(\hat{\mathbf{Q}})$ is the ball in $\mathbb{H}$ with center $\hat{\mathbf{Q}}$ and radius $r$ w.r.t. the Frobenius norm), but we will later apply it with $\mathbf{Q} = \hat{\mathbf{Q}}_N$ for some $N \in \mathbb{N}$.

### 3.3.1 Preliminary Notation and Definitions

The "directions" of the derivatives, which we define below, are elements in the unit sphere of the tangent space of $\mathbb{H}$, i.e.,

$$\mathbb{D} := \{\mathbf{D} \in \mathbb{R}^{D \times D} \mid \mathbf{D} = \mathbf{D}^T, \operatorname{tr}(\mathbf{D}) = 0, \|\mathbf{D}\|_F = 1\}.$$

Throughout the paper, directions in $\mathbb{D}$ are often determined by particular points $\mathbf{Q}_1, \mathbf{Q}_2 \in \mathbb{H}$, where $\mathbf{Q}_1 \neq \mathbf{Q}_2$. We denote the direction from $\mathbf{Q}_1$ to $\mathbf{Q}_2$ by $\mathbf{D}_{\mathbf{Q}_1, \mathbf{Q}_2}$, that is,

$$\mathbf{D}_{\mathbf{Q}_1, \mathbf{Q}_2} := \frac{\mathbf{Q}_2 - \mathbf{Q}_1}{\|\mathbf{Q}_2 - \mathbf{Q}_1\|_F}. \tag{3.2}$$

Directional derivatives with respect to an element of $\mathbb{D}$ may not exist and therefore we use directional derivatives from the right. That is, for $\mathbf{Q} \in \mathbb{H}$ and $\mathbf{D} \in \mathbb{D}$, the directional derivative (from the right) of $F$ at $\mathbf{Q}$ in the direction $\mathbf{D}$ is

$$\nabla_{\mathbf{D}}^+ F(\mathbf{Q}) := \frac{\mathrm{d}}{\mathrm{d}t} F(\mathbf{Q} + t\mathbf{D})\big|_{t=0^+}. \tag{3.3}$$

### 3.3.2 Mathematical Statement

We use the above notation to formulate the desired bound on $\|\mathbf{Q} - \hat{\mathbf{Q}}\|_F$. It involves the constant $\alpha_0$, which is also used in Theorem 1.1. The proof of this lemma clarifies the existence of $\alpha_0$, though it does not suggest an explicit approximation for it.

**Lemma 3.3.** *For $r > 0$ there exists a constant $\alpha_0 \equiv \alpha_0(r, \mu, D) > 0$ such that for all $\mathbf{Q} \in B_r(\hat{\mathbf{Q}}) \setminus \{\hat{\mathbf{Q}}\}$:*

$$\nabla_{\mathbf{D}_{\hat{\mathbf{Q}}, \mathbf{Q}}}^+ F(\mathbf{Q}) - \nabla_{\mathbf{D}_{\hat{\mathbf{Q}}, \mathbf{Q}}}^+ F(\hat{\mathbf{Q}}) \geq \alpha_0 \|\mathbf{Q} - \hat{\mathbf{Q}}\|_F \tag{3.4}$$

*and consequently*

$$\nabla^+_{\mathbf{D}_{\hat{\mathbf{Q}},\mathbf{Q}}} F(\mathbf{Q}) \geq \alpha_0 \|\mathbf{Q} - \hat{\mathbf{Q}}\|_F. \tag{3.5}$$

## 3.4 $N^{-1/2}$ Convergence of Directional Derivatives

We formulate the following convergence rate of the directional derivatives of $F_N$ from the right:

**Theorem 3.4.** *For* $\mathbf{Q} \in \mathbb{H}$ *and* $\mathbf{D} \in \mathbb{D}$,

$$\mathbb{P}\left(\left|\nabla^+_{\mathbf{D}} F(\mathbf{Q}) - \nabla^+_{\mathbf{D}} F_N(\mathbf{Q})\right| \geq N^{\epsilon - \frac{1}{2}}\right) \leq 2 \exp\left(\frac{-N^{2\epsilon}}{D \cdot R^2_\mu}\right). \tag{3.6}$$

It will be desirable to replace $\nabla^+_{\mathbf{D}} F(\mathbf{Q}) - \nabla^+_{\mathbf{D}} F_N(\mathbf{Q})$ in (3.6) with $\nabla^+_{\mathbf{D}} F(\mathbf{Q})$, though it is impossible in general. We will later use the following observation to implicitly obtain a result in this direction.

**Lemma 3.5.** *If* $\mathbf{Q} \in \mathbb{H} \setminus \{\hat{\mathbf{Q}}\}$, *then*

$$\nabla^+_{\mathbf{D}_{\hat{\mathbf{Q}},\mathbf{Q}}} F(\mathbf{Q}) \geq 0. \tag{3.7}$$

## 3.5 An Incomplete Idea for Proving Theorem 1.1

At this point we can outline the basic intuition behind the proof of Theorem 1.1. We assume for simplicity that $\hat{\mathbf{Q}}_N$ is u.d. Suppose, for the moment, that we could use (3.6) of Theorem 3.4 with $\mathbf{Q} := \hat{\mathbf{Q}}_N$. This is actually not mathematically sound, as we will discuss shortly, but if we could do it then we would have from (3.6) that

$$\mathbb{P}\left(\left|\nabla^+_{\mathbf{D}_{\hat{\mathbf{Q}},\hat{\mathbf{Q}}_N}} F(\hat{\mathbf{Q}}_N) - \nabla^+_{\mathbf{D}_{\hat{\mathbf{Q}},\hat{\mathbf{Q}}_N}} F_N(\hat{\mathbf{Q}}_N)\right| \geq N^{\epsilon - \frac{1}{2}}\right) \leq 2 \exp\left(\frac{-N^{2\epsilon}}{D \cdot R^2_\mu}\right). \tag{3.8}$$

We note that (3.7) as well as both the convexity of $F_N$ and the definition of $\hat{\mathbf{Q}}_N$ imply that

$$\nabla^+_{\mathbf{D}_{\hat{\mathbf{Q}},\hat{\mathbf{Q}}_N}} F(\hat{\mathbf{Q}}_N) \geq 0 \quad \text{and} \quad \nabla^+_{\mathbf{D}_{\hat{\mathbf{Q}},\hat{\mathbf{Q}}_N}} F_N(\hat{\mathbf{Q}}_N) \leq 0. \tag{3.9}$$

Combining (3.8) and (3.9), we obtain that

$$\mathbb{P}\left(\nabla^+_{\mathbf{D}_{\mathbf{Q},\hat{\mathbf{Q}}_N}} F(\hat{\mathbf{Q}}_N) \geq N^{\epsilon - \frac{1}{2}}\right) \leq 2 \exp\left(\frac{-N^{2\epsilon}}{D \cdot R^2_\mu}\right). \tag{3.10}$$

At last, combining (3.5), (3.10) and Theorem 3.2 we can formally prove Theorem 1.1.

However, as mentioned above, we cannot legally use Theorem 3.4 with $\mathbf{Q} = \hat{\mathbf{Q}}_N$. This is because $\hat{\mathbf{Q}}_N$ is a function of the samples (random variables) $\{\mathbf{x}_i\}^N_{i=1}$, but for our proof to be valid, $\mathbf{Q}$ needs to be fixed before the sampling begins.

Therefore, our new goal is to utilize the intuition described above, but modify the proof to make it mathematically sound. This is accomplished by creating a series of "nets" (subsets of $\mathbb{H}$) of increasing precision. Each matrix in each of the nets is determined before the sampling begins, so it can be used in Theorem 3.4. However, the construction of the nets guarantees that the $N$th net contains a matrix $\mathbf{Q}$ which is sufficiently close to $\hat{\mathbf{Q}}_N$ to be used as a substitute for $\hat{\mathbf{Q}}_N$ in the above process.

## 3.6 The Missing Component: Adaptive Nets

We describe here a result on the existence of a sequence of nets as suggested in §3.5. They are constructed in several stages, which cannot fit in here (see careful explanation in the extended version of this paper). We recall that $B_2(\hat{\mathbf{Q}})$ denotes a ball in $\mathbb{H}$ with center $\hat{\mathbf{Q}}$ and radius 2 w.r.t. the Frobenius norm.

**Lemma 3.6.** *Given* $\kappa \geq 2$ *and* $\tau > 0$, *there exists a sequence of sets* $\{S_n\}^\infty_{n=1}$ *such that* $\forall n \in \mathbb{N}$ $S_n \subset B_2(\hat{\mathbf{Q}})$ *and for any* $\mathbf{Q} \in B_2(\hat{\mathbf{Q}})$ *with* $\|\mathbf{Q} - \hat{\mathbf{Q}}\|_F > n^{-\frac{1}{2}}$, $\exists \mathbf{Q}' \in S_n$ *with*

$$\|\mathbf{Q}' - \hat{\mathbf{Q}}\|_F \leq \|\mathbf{Q} - \hat{\mathbf{Q}}\|_F, \tag{3.11}$$

$$2n^{-\frac{1}{2}}(\tau + \kappa^{-1}) \geq \|\mathbf{Q}' - \mathbf{Q}\|_F \geq n^{-\frac{1}{2}}\kappa^{-1} \quad and \tag{3.12}$$

$$\|\mathbf{D}_{\hat{\mathbf{Q}},\mathbf{Q}'} - \mathbf{D}_{\hat{\mathbf{Q}},\mathbf{Q}}\|_F \leq \tau n^{-1}. \tag{3.13}$$

*Furthermore,*

$$|S_n| \leq 2\kappa n^{\frac{1}{2}} \left( \frac{10Dn}{\tau} \right)^{\frac{D(D+1)}{2}}. \tag{3.14}$$

The following lemma shows that we can use $S_N$ to guarantee good approximation of $\hat{\mathbf{Q}}$ by $\hat{\mathbf{Q}}_N$ as long as the differences of partial derivatives are well-controlled for elements of $S_N$ (it uses the fixed constants $\kappa$ and $\tau$ for $S_N$; see Lemma 3.6).

**Lemma 3.7.** *If for some $\epsilon > 0$, $F_N$ is strictly convex and*

$$\left| \nabla^+_{\mathbf{D}_{\mathbf{Q},\hat{\mathbf{Q}}}} F(\mathbf{Q}) - \nabla^+_{\mathbf{D}_{\mathbf{Q},\hat{\mathbf{Q}}}} F_N(\mathbf{Q}) \right| \leq N^{-\frac{1}{2}+\epsilon} \quad \forall \mathbf{Q} \in S_N, \tag{3.15}$$

*then $\hat{\mathbf{Q}}_N$ is u.d. and*

$$\|\hat{\mathbf{Q}} - \hat{\mathbf{Q}}_N\|_F \leq \frac{1 + 2\alpha_0(\tau + \frac{1}{\kappa}) + 4R_\mu\kappa\tau}{\alpha_0} N^{-\frac{1}{2}+\epsilon}. \tag{3.16}$$

### 3.7 Completing the Proof of Theorem 1.1

Let us fix $\kappa_0 = (4\alpha_0) \vee 2$, $\tau_0 := (1 - 2\alpha_0/\kappa_0)/(2\alpha_0 + 4R_\mu\kappa_0)$ and $N > 2(D - 1)$. We note that

$$1 + 2\alpha_0(\tau_0 + \frac{1}{\kappa_0}) + 4R_\mu\kappa_0\tau_0 = 2. \tag{3.17}$$

We rewrite (3.14) using $\kappa := \kappa_0$ and $\tau := \tau_0$ and then bound its RHS from above as follows

$$|S_N| \leq 2((4\alpha_0) \vee 2)N^{\frac{D^2+D+1}{2}} \left( 10D\frac{2\alpha_0 + 4R_\mu((4\alpha_0) \vee 2)}{1 - \frac{2\alpha_0}{(4\alpha_0)\vee 2}} \right)^{\frac{D(D+1)}{2}} \tag{3.18}$$

$$\leq \frac{C_0}{2} N^{D^2}.$$

Combining (3.6) (applied to any $\mathbf{Q} \in S_N$) and (3.18) we obtain that

$$\mathbb{P}\left( \exists \mathbf{Q} \in S_N \text{ with } \left| \nabla^+_{\mathbf{D}_{\mathbf{Q},\hat{\mathbf{Q}}}} F(\mathbf{Q}) - \nabla^+_{\mathbf{D}_{\mathbf{Q},\hat{\mathbf{Q}}}} F_N(\mathbf{Q}) \right| \geq N^{-\frac{1}{2}+\epsilon} \right)$$
$$\leq C_0 N^{D^2} \exp\left( -N^{2\epsilon}/(D \cdot R_\mu^2) \right). \tag{3.19}$$

Furthermore, (3.1) and (3.19) imply that

$$\mathbb{P}\left( \left| \nabla^+_{\mathbf{D}_{\mathbf{Q},\hat{\mathbf{Q}}}} F(\mathbf{Q}) - \nabla^+_{\mathbf{D}_{\mathbf{Q},\hat{\mathbf{Q}}}} F_N(\mathbf{Q}) \right| \leq N^{-\frac{1}{2}+\epsilon} \ \forall \mathbf{Q} \in S_N \text{ and } \hat{\mathbf{Q}}_N \text{ is u.d.} \right)$$
$$\geq 1 - C_0 N^{D^2} \exp\left( \frac{-N^{2\epsilon}}{D \cdot R_\mu^2} \right) - 2\binom{N}{D-1}^2 (1-\gamma)^{N-2(D-1)}. \tag{3.20}$$

Theorem 1.1 clearly concludes from Lemma 3.7 (applied with $\kappa := \kappa_0$ and $\tau := \tau_0$), (3.20) and (3.17).

### Acknowledgment

This work was supported by NSF grants DMS-09-15064 and DMS-09-56072. Part of this work was performed when M. Coudron attended the University of Minnesota (as an undergraduate student). We thank T. Zhang for valuable conversations and forwarding us [22].

# References

[1] A. Agarwal, S. Negahban, and M. Wainwright. Fast global convergence of gradient methods for high-dimensional statistical recovery. Technical Report arXiv:1104.4824, Apr 2011.

[2] A. Agarwal, S. Negahban, and M. Wainwright. Noisy matrix decomposition via convex relaxation: Optimal rates in high dimensions. In *ICML*, pages 1129–1136, 2011.

[3] T. T. Cai, C.-H. Zhang, and H. H. Zhou. Optimal rates of convergence for covariance matrix estimation. *Ann. Statist.*, 38(4):2118–2144, 2010.

[4] E. J. Candès, X. Li, Y. Ma, and J. Wright. Robust principal component analysis? *J. ACM*, 58(3):11, 2011.

[5] V. Chandrasekaran, S. Sanghavi, P. A. Parrilo, and A. S. Willsky. Rank-sparsity incoherence for matrix decomposition. *Arxiv*, 02139:1–24, 2009.

[6] C. Davis and W. M. Kahan. The rotation of eigenvectors by a perturbation. iii. *SIAM J. on Numerical Analysis*, 7:1–46, 1970.

[7] D. Hsu, S. Kakade, and T. Zhang. Robust matrix decomposition with sparse corruptions. *Information Theory, IEEE Transactions on*, 57(11):7221 –7234, nov. 2011.

[8] P. J. Huber and E. Ronchetti. *Robust statistics*. Wiley series in probability and mathematical statistics. Probability and mathematical statistics. Wiley, 2009.

[9] G. Lerman, M. McCoy, J. A. Tropp, and T. Zhang. Robust computation of linear models, or How to find a needle in a haystack. *ArXiv e-prints*, Feb. 2012.

[10] R. A. Maronna, R. D. Martin, and V. J. Yohai. *Robust statistics: Theory and methods*. Wiley Series in Probability and Statistics. John Wiley & Sons Ltd., Chichester, 2006.

[11] M. McCoy and J. Tropp. Two proposals for robust PCA using semidefinite programming. *Elec. J. Stat.*, 5:1123–1160, 2011.

[12] P. Ravikumar, M. J. Wainwright, G. Raskutti, and B. Yu. High-dimensional covariance estimation by minimizing $\ell_1$-penalized log-determinant divergence. *Electron. J. Stat.*, 5:935–980, 2011.

[13] A. J. Rothman, P. J. Bickel, E. Levina, and J. Zhu. Sparse permutation invariant covariance estimation. *Electron. J. Stat.*, 2:494–515, 2008.

[14] P. J. Rousseeuw and A. M. Leroy. *Robust regression and outlier detection*. Wiley Series in Probability and Mathematical Statistics: Applied Probability and Statistics. John Wiley & Sons Inc., New York, 1987.

[15] J. Shawe-taylor, C. Williams, N. Cristianini, and J. Kandola. On the eigenspectrum of the Gram matrix and the generalisation error of kernel PCA. *IEEE Transactions on Information Theory*, 51(1):2510–2522, 2005.

[16] L. G. Valiant. A theory of the learnable. *Commun. ACM*, 27(11):1134–1142, Nov. 1984.

[17] R. Vershynin. How close is the sample covariance matrix to the actual covariance matrix? to appear.

[18] R. Vershynin. Introduction to the non-asymptotic analysis of random matrices. In Y. C. Eldar and G. Kutyniok, editors, *Compressed Sensing: Theory and Applications*. Cambridge Univ Press, to appear.

[19] H. Xu, C. Caramanis, and S. Sanghavi. Robust pca via outlier pursuit. In *NIPS*, pages 2496–2504, 2010.

[20] H. Xu, C. Caramanis, and S. Sanghavi. Robust pca via outlier pursuit. *Information Theory, IEEE Transactions on*, PP(99):1, 2012.

[21] T. Zhang and G. Lerman. A novel m-estimator for robust pca. Submitted, available at arXiv:1112.4863.

[22] T. Zhang and H. Zou. Sparse precision matrix estimation via positive definite constrained minimization of $\ell_1$ penalized d-trace loss. Personal Communication, 2012.

